# Autonomous Learning of Action Models for Planning

**Neville Mehta**      **Prasad Tadepalli**      **Alan Fern**
School of Electrical Engineering and Computer Science
Oregon State University, Corvallis, OR 97331, USA.
{mehtane,tadepall,afern}@eecs.oregonstate.edu

## Abstract

This paper introduces two new frameworks for learning action models for planning. In the mistake-bounded planning framework, the learner has access to a planner for the given model representation, a simulator, and a planning problem generator, and aims to learn a model with at most a polynomial number of faulty plans. In the planned exploration framework, the learner does not have access to a problem generator and must instead design its own problems, plan for them, and converge with at most a polynomial number of planning attempts. The paper reduces learning in these frameworks to concept learning with one-sided error and provides algorithms for successful learning in both frameworks. A specific family of hypothesis spaces is shown to be efficiently learnable in both the frameworks.

## 1  Introduction

Planning research typically assumes that the planning system is provided complete and correct models of the actions. However, truly autonomous agents must learn these models. Moreover, model learning, planning, and plan execution must be interleaved, because agents need to plan long before perfect models are learned. This paper formulates and analyzes the learning of deterministic action models used in planning for goal achievement. It has been shown that deterministic STRIPS actions with a constant number of preconditions can be learned from raw experience with at most a polynomial number of plan prediction mistakes [8]. In spite of this positive result, compact action models in fully observable, deterministic action models are not always efficiently learnable. For example, action models represented as arbitrary Boolean functions are not efficiently learnable under standard cryptographic assumptions such as the hardness of factoring [2].

Learning action models for planning is different from learning an arbitrary function from states and actions to next states, because one can ignore modeling the effects of some actions in certain contexts. For example, most people who drive do not ever learn a complete model of the dynamics of their vehicles; while they might accurately know the stopping distance or turning radius, they could be oblivious to many aspects that an expert auto mechanic is comfortable with. To capture this intuition, we introduce the concept of an *adequate* model, that is, a model that is sound and sufficiently complete for planning for a given class of goals. When navigating a city, any spanning tree of the transportation network connecting the places of interest would be an adequate model.

We define two distinct frameworks for learning adequate models for planning and then characterize sufficient conditions for success in these frameworks. In the *mistake-bounded planning* (MBP) framework, the goal is to continually solve user-generated planning problems while learning action models and guarantee at most a polynomial number of faulty plans or mistakes. We assume that in addition to the problem generator, the learner has access to a sound and complete planner and a simulator (or the real world). We also introduce a more demanding *planned exploration* (PLEX) framework, where the learner needs to generate its own problems to refine its action model. This requirement translates to an experiment-design problem, where the learner needs to design problems in a goal language to refine the action models.

The MBP and PLEX frameworks can be reduced to *over-general query* learning, concept learning with strictly one-sided error, where the learner is only allowed to make false positive mistakes [7]. This is ideally suited for the autonomous learning setting in which there is no oracle who can provide positive examples of plans or demonstrations, but negative examples are observed when the agent's plans fail to achieve their goals. We introduce mistake-bounded and exact learning versions of this learning framework and show that they are strictly more powerful than the recently introduced KWIK framework [4]. We view an action model as a set of state-action-state transitions and ensure that the learner always maintains a hypothesis which includes all transitions in some adequate model. Thus, a sound plan is always in the learner's search space, while it may not always be generated. As the learner gains more experience in generating plans, executing them on the simulator, and receiving observations, the hypothesis is incrementally refined until an adequate model is discovered. To ground our analysis, we show that a general family of hypothesis spaces is learnable in polynomial time in the two frameworks given appropriate goal languages. This family includes a generalization of propositional STRIPS operators with conditional effects.

## 2 Over-General Query Learning

We first introduce a variant of a concept-learning framework that serves as formal underpinning of our model-learning frameworks. This variant is motivated by the principle of "optimism under uncertainty", which is at the root of several related algorithms in reinforcement learning [1, 3].

A *concept* is a set of instances. An *hypothesis space* $\mathcal{H}$ is a set of strings or hypotheses, each of which represents a concept. The size of the concept is the length of the smallest hypothesis that represents it. Without loss of generality, $\mathcal{H}$ can be structured as a (directed acyclic) *generalization graph*, where the nodes correspond to sets of equivalent hypotheses representing a concept and there is a directed edge from node $n_1$ to node $n_2$ if and only if the concept at $n_1$ is strictly more general than (a strict superset of) that at $n_2$.

**Definition 2.1.** *The* height *of $\mathcal{H}$ is a function of $n$ and is the length of the longest path from a root node to any node representing concepts of size $n$ in the generalization graph of $\mathcal{H}$.*

**Definition 2.2.** *A hypothesis $h$ is* consistent *with a set of negative examples $Z$ if $h \cap Z = \varnothing$. Given a set of negative examples $Z$ consistent with a target hypothesis $h$, the* version space *of action models is the subset of all hypotheses in $\mathcal{H}$ that are consistent with $Z$ and is denoted as $\mathcal{M}(Z)$.*

**Definition 2.3.** *$\mathcal{H}$ is* well-structured *if, for any negative example set $Z$ which has a consistent target hypothesis in $\mathcal{H}$, the version space $\mathcal{M}(Z)$ contains a most general hypothesis $mgh(Z)$. Further, $\mathcal{H}$ is* efficiently well-structured *if there exists an algorithm that can compute $mgh(Z \cup \{z\})$ from $mgh(Z)$ and a new example $z$ in time polynomial in the size of $mgh(Z)$ and $z$.*

**Lemma 2.1.** *Any finite hypothesis space $\mathcal{H}$ is well-structured if and only if it is closed under union.*
*Proof.* (If) Let $Z$ be a set of negative examples and let $H_0 = \bigcup_{h \in \mathcal{M}(Z)} h$ represent the unique union of all concepts represented by hypotheses in $\mathcal{M}(Z)$. Because $\mathcal{H}$ is closed under union and finite, $H_0$ must be in $\mathcal{H}$. If $\exists z \in H_0 \cap Z$, then $z \in h \cap Z$ for some $h \in \mathcal{M}(Z)$. This is a contradiction, because all $h \in \mathcal{M}(Z)$ are consistent with $Z$. Consequently, $H_0$ is consistent with $Z$, and is in $\mathcal{M}(Z)$. It is more general than (is a superset of) every other hypothesis in $\mathcal{M}(Z)$ because it is their union.

(Only if) Let $h_1, h_2$ be any two hypotheses in $\mathcal{H}$ and $Z$ be the set of all instances not included in either $h_1$ and $h_2$. Both $h_1$ and $h_2$ are consistent with examples in $Z$. As $\mathcal{H}$ is well-structured, $mgh(Z)$ must also be in the version space $\mathcal{M}(Z)$, and consequently in $\mathcal{H}$. However, $mgh(Z) = h_1 \cup h_2$ because it cannot include any element without $h_1 \cup h_2$ and must include all elements within. Hence, $h_1 \cup h_2$ is in $\mathcal{H}$, which makes it closed under union. $\square$

In the *over-general query* (OGQ) framework, the teacher selects a target concept $c \in \mathcal{H}$. The learner outputs a query in the form of a hypothesis $h \in \mathcal{H}$, where $h$ must be at least as general as $c$. The teacher responds with yes if $h \equiv c$ and the episode ends; otherwise, the teacher gives a counterexample $x \in h - c$. The learner then outputs a new query, and the cycle repeats.

**Definition 2.4.** *A hypothesis space is* OGQ-learnable *if there exists a learning algorithm for the OGQ framework that identifies the target $c$ with the number of queries and total running time that is polynomial in the size of $c$ and the size of the largest counterexample.*

**Theorem 1.** *$\mathcal{H}$ is learnable in the OGQ framework if and only if $\mathcal{H}$ is efficiently well-structured and its height is a polynomial function.*

*Proof.* (If) If $\mathcal{H}$ is efficiently well-structured, then the OGQ learner can always output the mgh, guaranteed to be more general than the target concept, in polynomial time. Because the maximum number of hypothesis refinements is bounded by the polynomial height of $\mathcal{H}$, it is learnable in the OGQ framework.

(Only if) If $\mathcal{H}$ is not well-structured, then $\exists h_1, h_2 \in \mathcal{H}, h_1 \cup h_2 \notin \mathcal{H}$. The teacher can delay picking its target concept, but always provide counterexamples from outside both $h_1$ and $h_2$. At some point, these counterexamples will force the learner to choose between $h_1$ or $h_2$, because their union is not in the hypothesis space. Once the learner makes its choice, the teacher can choose the other hypothesis as its target concept $c$, resulting in the learner's hypothesis not being more general than $c$. If $\mathcal{H}$ is not efficiently well-structured, then there exists $Z$ and $z$ such that computing $\mathrm{mgh}(Z \cup \{z\})$ from $\mathrm{mgh}(Z)$ and a new example $z$ cannot be done in polynomial time. If the teacher picks $\mathrm{mgh}(Z \cup \{z\})$ as the target concept and only provides counterexamples from $Z \cup \{z\}$, then the learner cannot have polynomial running time. Finally, the teacher can always provide counterexamples that forces the learner to take the longest path in $\mathcal{H}$'s generalization graph. Thus, if $\mathcal{H}$ does not have polynomial height, then the number of queries will not be polynomial. $\square$

## 2.1 A Comparison of Learning Frameworks

In order to compare the OGQ framework to other learning frameworks, we first define the *over-general mistake-bounded* (OGMB) learning framework, in which the teacher selects a target concept $c$ from $\mathcal{H}$ and presents an arbitrary instance $x$ from the instance space to the learner for a prediction. An inclusion mistake is made when the learner predicts $x \in c$ although $x \notin c$; an exclusion mistake is made when the learner predicts $x \notin c$ although $x \in c$. The teacher presents the true label to the learner if a mistake is made, and then presents the next instance to the learner, and so on.

**Definition 2.5.** *A hypothesis space is* OGMB-learnable *if there exists a learning algorithm for the OGMB framework that never makes any exclusion mistakes and its number of inclusion mistakes and the running time on each instance are both bounded by polynomial functions of the size of the target concept and the size of the largest instance seen by the learner.*

In the following analysis, we let the name of a framework denote the set of hypothesis spaces learnable in that framework.

**Theorem 2.** *OGQ $\subsetneq$ OGMB.*

*Proof.* We can construct an OGMB learner from the OGQ learner as follows. When the OGQ learner makes a query $h$, we use $h$ to make predictions for the OGMB learner. As $h$ is guaranteed to be over-general, it never makes an exclusion mistake. Any instance $x$ on which it makes an inclusion mistake must be in $h - c$ and this is returned to the OGQ learner. The cycle repeats with the OGQ learner providing a new query. Because the OGQ learner makes only a polynomial number of queries and takes polynomial time for query generation, the simulated OGMB learner makes only a polynomial number of mistakes and runs in at most polynomial time per instance. The converse does not hold in general because the queries of the OGQ learner are restricted to be "proper", that is, they must belong to the given hypothesis space. While the OGMB learner can maintain the version space of all consistent hypotheses of a polynomially-sized hypothesis space, the OGQ learner can only query with a single hypothesis and there may not be any hypothesis that is guaranteed to be more general than the target concept. $\square$

If the learner is allowed to ask queries outside $\mathcal{H}$, such as queries of the form $h_1 \cup \cdots \cup h_n$ for all $h_i$ in the version space, then over-general learning is possible. In general, if the learner is allowed to ask about any polynomially-sized, polynomial-time computable hypothesis, then it is as powerful as OGMB, because it can encode the computation of the OGMB learner inside a polynomial-sized circuit and query with that as the hypothesis. We call this the OGQ+ framework and claim the following theorem (the proof is straightforward).

**Theorem 3.** *OGQ+ = OGMB.*

The Knows-What-It-Knows (KWIK) learning framework [4] is similar to the OGMB framework with one key difference: it does not allow the learner to make any prediction when it does not know the correct answer. In other words, the learner either makes a correct prediction or simply abstains from making a prediction and gets the true label from the teacher. The number of abstentions is

bounded by a polynomial in the target size and the largest instance size. The set of hypothesis spaces learnable in the mistake-bound (MB) framework is a strict subset of that learnable in the probably-approximately-correct (PAC) framework [5], leading to the following result.

**Theorem 4.** *KWIK $\subsetneq$ OGMB $\subsetneq$ MB $\subsetneq$ PAC.*

*Proof.* OGMB $\subsetneq$ MB: Every hypothesis space that is OGMB-learnable is MB-learnable because the OGMB learner is additionally constrained to not make an exclusion mistake. However, every MB-learnable hypothesis space is not OGMB-learnable. Consider the hypothesis space of conjunctions of $n$ Boolean literals (positive or negative). A single exclusion mistake is sufficient for an MB learner to learn this hypothesis space. In contrast, after making an inclusion mistake, the OGMB learner can only exclude that example from the candidate set. As there is exactly one positive example, this could force the OGMB learner to make an exponential number of mistakes (similar to guessing an unknown password).

KWIK $\subsetneq$ OGMB: If a concept class is KWIK-learnable, it is also OGMB-learnable — when the KWIK learner does not know the true label, the OGMB learner simply predicts that the instance is positive and gets corrected if it is wrong. However, every OGMB-learnable hypothesis space is not KWIK-learnable. Consider the hypothesis space of disjunctions of $n$ Boolean literals. The OGMB learner begins with a disjunction over all possible literals (both positive and negative) and hence predicts all instances as positive. A single inclusion mistake is sufficient for the OGMB learner to learn this hypothesis space. On the other hand, the teacher can supply the KWIK learner with an exponential number of positive examples, because the KWIK learner cannot ever know that the target does not include all possible instances; this implies that the number of abstentions is not polynomially bounded. $\square$

This theorem demonstrates that KWIK is too conservative a framework for model learning — any prediction that might be a mistake is disallowed. This makes it impossible to learn even simple concept classes such as pure disjunctions.

## 3 Planning Components

A factored planning domain $\mathcal{P}$ is a tuple $(V, D, A, T)$, where $V = \{v_1, \ldots, v_n\}$ is the set of variables, $D$ is the domain of the variables in $V$, and $A$ is the set of actions. $S = D^n$ represents the state space and $T \subset S \times A \times S$ is the transition relation, where $(s, a, s') \in T$ signifies that taking action $a$ in state $s$ results in state $s'$. As we only consider learning deterministic action models, the transition relation is in fact a function, although the learner's hypothesis space may include nondeterministic models. The domain parameters, $n, |D|$, and $|A|$, characterize the size of $\mathcal{P}$ and are *implicit in all claims of complexity* in the rest of this paper.

**Definition 3.1.** *An* action model *is a relation $M \subseteq S \times A \times S$.*

A *planning problem* is a pair $(s_0, g)$, where $s_0 \in S$ and the goal condition $g$ is an expression chosen from a goal language $\mathcal{G}$ and represents a set of states in which it evaluates to true. A state $s$ *satisfies* a goal $g$ if and only if $g$ is true in $s$. Given a planning problem $(s_0, g)$, a *plan* is a sequence of states and actions $s_0, a_1, \ldots, a_p, s_p$, where the state $s_p$ satisfies the goal $g$. The plan is *sound* with respect to $(M, g)$ if $(s_{i-1}, a_i, s_i) \in M$ for $1 \leq i \leq p$.

**Definition 3.2.** *A* planner *for the hypothesis space and goal language pair $(\mathcal{H}, \mathcal{G})$ is an algorithm that takes $M \in \mathcal{H}$ and $(s_0, g \in \mathcal{G})$ as inputs and outputs a plan or signals failure. It is* sound *with respect to $(\mathcal{H}, \mathcal{G})$ if, given any $M$ and $(s_0, g)$, it produces a sound plan with respect to $(M, g)$ or signals failure. It is* complete *with respect to $(\mathcal{H}, \mathcal{G})$ if, given any $M$ and $(s_0, g)$, it produces a sound plan whenever one exists with respect to $(M, g)$.*

We generalize the definition of soundness from its standard usage in the literature in order to apply to nondeterministic action models, where the nondeterminism is "angelic" — the planner can control the outcome of actions when multiple outcomes are possible according to its model [6]. One way to implement such a planner is to do forward search through all possible action and outcome sequences and return an action sequence if it leads to a goal under some outcome choices. Our analysis is agnostic to plan quality or plan length and applies equally well to suboptimal planners. This is motivated by the fact that optimal planning is hard for most domains, but suboptimal planning such as hierarchical planning can be quite efficient and practical.

**Definition 3.3.** *A planning mistake occurs if either the planner signals failure when a sound plan exists with respect to the transition function $T$ or when the plan output by the planner is not sound with respect to $T$.*

**Definition 3.4.** *Let $\mathcal{P}$ be a planning domain and $\mathcal{G}$ be a goal language. An action model $M$ is* adequate *for $\mathcal{G}$ in $\mathcal{P}$ if $M \subseteq T$ and the existence of a sound plan with respect to $(T, g \in \mathcal{G})$ implies the existence of a sound plan with respect to $(M, g)$. $\mathcal{H}$ is adequate for $\mathcal{G}$ if $\exists M \in \mathcal{H}$ such that $M$ is adequate for $\mathcal{G}$.*

An adequate model may be partial or incomplete in that it may not include every possible transition in the transition function $T$. However, the model is sufficient to produce a sound plan with respect to $(T, g)$ for every goal $g$ in the desired language. Thus, the more limited the goal language, the more incomplete the adequate model can be. In the example of a city map, if the goal language excludes certain locations, then the spanning tree could exclude them as well, although not necessarily so.

**Definition 3.5.** *A* simulator *of the domain is always situated in the current state $s$. It takes an action $a$ as input, transitions to the state $s'$ resulting from executing $a$ in $s$, and returns the current state $s'$. Given a goal language $\mathcal{G}$, a* problem generator *generates an arbitrary problem $(s_0, g \in \mathcal{G})$ and sets the state of the simulator to $s_0$.*

## 4  Mistake-Bounded Planning Framework

This section constructs the MBP framework that allows learning and planning to be interleaved for user-generated problems. It actualizes the teacher of the OGQ framework by combining a problem generator, a planner, and a simulator, and interfaces with the OGQ learner to learn action models as hypotheses over the space of possible state transitions for each action. It turns out that the one-sided mistake property is needed for autonomous learning because the learner can only learn by generating plans and observing the results; if the learner ever makes an exclusion error, there is no guarantee of finding a sound plan even when one exists and the learner cannot recover from such mistakes.

**Definition 4.1.** *Let $\mathcal{G}$ be a goal language such that $\mathcal{H}$ is adequate for it. $\mathcal{H}$ is* learnable in the MBP framework *if there exists an algorithm $\mathcal{A}$ that interacts with a problem generator over $\mathcal{G}$, a sound and complete planner with respect to $(\mathcal{H}, \mathcal{G})$, and a simulator of the planning domain $\mathcal{P}$, and outputs a plan or signals failure for each planning problem while guaranteeing at most a polynomial number of planning mistakes. Further, $\mathcal{A}$ must respond in time polynomial in the domain parameters and the length of the longest plan generated by the planner, assuming that a call to the planner, simulator, or problem generator takes $O(1)$ time.*

The goal language is picked such that the hypothesis space is adequate for it. We cannot bound the time for the convergence of $\mathcal{A}$, because there is no limit on when the mistakes are made.

**Theorem 5.** *$\mathcal{H}$ is learnable in the MBP framework if $\mathcal{H}$ is OGQ-learnable.*

*Proof.* Algorithm 1 is a general schema for action model learning in the MBP framework. The model $M$ begins with the initial query from OGQ-LEARNER. PROBLEMGENERATOR provides a planning problem and initializes the current state of SIMULATOR. Given $M$ and the planning problem, PLANNER always outputs a plan if one exists because $\mathcal{H}$ is adequate for $\mathcal{G}$ (it contains a "target" adequate model) and $M$ is at least as general as every adequate model. If PLANNER signals failure, then there is no plan for it. Otherwise, the plan is executed through SIMULATOR until an observed transition conflicts with the predicted transition. If such a transition is found, it is supplied to OGQ-LEARNER and $M$ is updated with the next query; otherwise, the plan is output. If $\mathcal{H}$ is OGQ-learnable, then OGQ-LEARNER will only be called a polynomial number of times, every call taking polynomial time. As the number of planning mistakes is

---

**Algorithm 1** MBP LEARNING SCHEMA

**Input**: Goal language $\mathcal{G}$
1: $M \leftarrow$ OGQ-LEARNER()     // Initial query
2: **loop**
3:     $(s, g) \leftarrow$ PROBLEMGENERATOR$(\mathcal{G})$
4:     $plan \leftarrow$ PLANNER$(M, (s, g))$
5:     **if** $plan \neq$ **false then**
6:         **for** $(\hat{s}, a, \hat{s}')$ **in** $plan$ **do**
7:             $s' \leftarrow$ SIMULATOR$(a)$
8:             **if** $s' \neq \hat{s}'$ **then**
9:                 $M \leftarrow$ OGQ-LEARNER$((s, a, \hat{s}'))$
10:                **print** mistake
11:                **break**
12:            $s \leftarrow s'$
13:        **if** no mistake **then**
14:            **print** $plan$

polynomial and every response of Algorithm 1 is polynomial in the runtime of OGQ-LEARNER and the length of the longest plan, $\mathcal{H}$ is learnable in the MBP framework. $\qquad\square$

The above result generalizes the work on learning STRIPS operator models from raw experience (without a teacher) in [8] to arbitrary hypotheses spaces by identifying sufficiency conditions. (A family of hypothesis spaces considered later in this paper subsumes propositional STRIPS by capturing conditional effects.) It also clarifies the notion of an adequate model, which can be much simpler than the true transition model, and the influence of the goal language on the complexity of learning action models.

## 5   Planned Exploration Framework

The MBP framework is appropriate when mistakes are permissible on user-given problems as long as their total number is limited and not for cases where no mistakes are permitted after the training period. In the planned exploration (PLEX) framework, the agent seeks to learn an action model for the domain without an external problem generator by generating planning problems for itself. The key issue here is to generate a reasonably small number of planning problems such that solving them would identify a deterministic action model. Learning a model in the PLEX framework involves knowing where it is deficient and then planning to reach states that are informative, which entails formulating planning problems in a goal language. This framework provides a polynomial sample convergence guarantee which is stronger than a polynomial mistake bound of the MBP framework. Without a problem generator that can change the simulator's state, it is impossible for the simulator to transition freely between strongly connected components (SCCs) of the transition graph. Hence, we make the assumption that the transition graph is a disconnected union of SCCs and require only that the agent learn the model for a single SCC that contains the initial state of the simulator.

**Definition 5.1.** *Let $\mathcal{P}$ be a planning domain whose transition graph is a union of SCCs. $(\mathcal{H}, \mathcal{G})$ is* learnable in the PLEX framework *if there exists an algorithm $\mathcal{A}$ that interacts with a sound and complete planner with respect to $(\mathcal{H}, \mathcal{G})$ and the simulator for $\mathcal{P}$ and outputs a model $M \in \mathcal{H}$ that is adequate for $\mathcal{G}$ within the SCC that contains the initial state $s_0$ of the simulator after a polynomial number of planning attempts. Further, $\mathcal{A}$ must run in polynomial time in the domain parameters and the length of the longest plan output by the planner, assuming that every call to the planner and the simulator takes $O(1)$ time.*

A key step in planned exploration is designing appropriate planning problems. We call these *experiments* because the goal of solving these problems is to disambiguate nondeterministic action models. In particular, the agent tries to reach an *informative* state where the current model is nondeterministic.

**Definition 5.2.** *Given a model $M$, the set of* informative states *is $I(M) = \{s : (s, a, s'), (s, a, s'') \in M \wedge s' \neq s''\}$, where $a$ is said to be* informative *in $s$.*

**Definition 5.3.** *A set of goals $G$ is a* cover *of a set of states $R$ if $\bigcup_{g \in G}\{s : s \text{ satisfies } g\} = R$.*

Given the goal language $\mathcal{G}$ and a model $M$, the problem of experiment design is to find a set of goals $G \subseteq \mathcal{G}$ such that the sets of states that satisfy the goals in $G$ collectively cover all informative states $I(M)$. If it is possible to plan to achieve one of these goals, then either the plan passes through a state where the model is nondeterministic or it executes successfully and the agent reaches the final goal state; in either case, an informative action can be executed and the observed transition is used to refine the model. If none of the goals in $G$ can be successfully planned for, then no informative states for that action are reachable. We formalize these intuitions below.

**Definition 5.4.** *The* width *of $(\mathcal{H}, \mathcal{G})$ is defined as $\max_{M \in \mathcal{H}} \min_{G \subseteq \mathcal{G}: G \text{ is a cover of } I(M)} |G|$, where $\min_G |G| = \infty$ if there is no $G \subseteq \mathcal{G}$ to cover a nonempty $I(M)$.*

**Definition 5.5.** *$(\mathcal{H}, \mathcal{G})$ permits* efficient *experiment design if, for any $M \in \mathcal{H}$, ① there exists an algorithm (EXPERIMENTDESIGN) that takes $M$ and $\mathcal{G}$ as input and outputs a polynomial-sized cover of $I(M)$ in polynomial time and ② there exists an algorithm (INFOACTIONSTATES) that takes $M$ and a state $s$ as input and outputs an informative action and two (distinct) predicted next states according to $M$ in polynomial time.*

If $(\mathcal{H}, \mathcal{G})$ permits efficient experiment design, then it has polynomial width because no algorithm can always guarantee to output a polynomial-sized cover otherwise.

**Theorem 6.** $(\mathcal{H}, \mathcal{G})$ *is learnable in the PLEX framework if it permits efficient experiment design, and $\mathcal{H}$ is adequate for $\mathcal{G}$ and is OGQ-learnable.*

*Proof.* Algorithm 2 is a general schema for action model learning in the PLEX framework. The model $M$ begins with the initial query from OGQ-LEARNER. Given $M$ and $\mathcal{G}$, EXPERIMENTDESIGN computes a polynomial-sized cover $G$. If $G$ is empty, then the model cannot be refined further; otherwise, given $M$ and a goal $g \in G$, PLANNER may signal failure if either no state satisfies $g$ or states satisfying $g$ are not reachable from the current state of the simulator. If PLANNER signals failure on all of the goals, then none of the informative states are reachable and $M$ cannot be refined further. If PLANNER does output a plan, then the plan is executed through SIMULATOR until an observed transition conflicts with the predicted transition. If such a transition is found, it is supplied to OGQ-LEARNER and $M$ is updated with the next query. If the plan executes successfully, then INFOACTIONSTATES provides an informative action with the corresponding set of two resultant states according to $M$; OGQ-LEARNER is supplied with the transition of the goal state, the informative action, and the incor-

---

**Algorithm 2** PLEX LEARNING SCHEMA

**Input**: Initial state $s$, goal language $\mathcal{G}$
**Output**: Model $M$
1: $M \leftarrow$ OGQ-LEARNER()    // Initial query
2: **loop**
3:     $G \leftarrow$ EXPERIMENTDESIGN$(M, \mathcal{G})$
4:     **if** $G = \varnothing$ **then**
5:         **return** $M$
6:     **for** $g \in G$ **do**
7:         $plan \leftarrow$ PLANNER$(M, (s, g))$
8:         **if** $plan \neq$ **false then**
9:             **break**
10:     **if** $plan =$ **false then**
11:         **return** $M$
12:     **for** $(\hat{s}, a, \hat{s}')$ **in** $plan$ **do**
13:         $s' \leftarrow$ SIMULATOR$(a)$
14:         $s \leftarrow s'$
15:         **if** $s' \neq \hat{s}'$ **then**
16:             $M \leftarrow$ OGQ-LEARNER$((s, a, \hat{s}'))$
17:             **break**
18:     **if** $M$ has not been updated **then**
19:         $(a, \hat{S}') \leftarrow$ INFOACTIONSTATES$(M, s)$
20:         $s' \leftarrow$ SIMULATOR$(a)$
21:         $M \leftarrow$ OGQ-LEARNER$((s, a, \hat{s}' \in \hat{S}' - \{s'\}))$
22:         $s \leftarrow s'$
23: **return** $M$

---

rectly predicted next state, and $M$ is updated with the new query. A new cover is computed every time $M$ is updated, and the process continues until all experiments are exhausted. If $(\mathcal{H}, \mathcal{G})$ permits efficient experiment design, then every cover can be computed in polynomial time and INFOACTIONSTATES is efficient. If $\mathcal{H}$ is OGQ-learnable, then OGQ-LEARNER will only be called a polynomial number of times and it can output a new query in polynomial time. As the number of failures per successful plan is bounded by a polynomial in the width $w$ of $(\mathcal{H}, \mathcal{G})$, the total number of calls to PLANNER is polynomial. Further, as the innermost loop of Algorithm 2 is bounded by the longest length $l$ of a plan, its running time is a polynomial in the domain parameters and $l$. Thus, $(\mathcal{H}, \mathcal{G})$ is learnable in the PLEX framework. $\qquad \square$

## 6  A Hypothesis Family for Action Modeling

This section proves the learnability of a hypothesis-space family for action modeling in the MBP and PLEX frameworks. Let $\mathcal{U} = \{u_1, u_2, \ldots\}$ be a polynomial-sized set of polynomially computable basis hypotheses (polynomial in the relevant parameters), where $u_i$ represents a deterministic set of transition tuples. Let Power$(\mathcal{U}) = \{\bigcup_{u \in H} u : H \subseteq \mathcal{U}\}$ and Pairs$(\mathcal{U}) = \{u_1 \cup u_2 : u_1, u_2 \in \mathcal{U}\}$.

**Lemma 6.1.** *Power$(\mathcal{U})$ is OGQ-learnable.*

*Proof.* Power$(\mathcal{U})$ is efficiently well-structured, because it is closed under union by definition and the new mgh can be computed by removing any basis hypotheses that are not consistent with the counterexample; this takes polynomial time as $\mathcal{U}$ is of polynomial size. At the root of the generalization graph of Power$(\mathcal{U})$ is the hypothesis $\bigcup_{u \in \mathcal{U}} u$ and at the leaf is the empty hypothesis. Because $\mathcal{U}$ is of polynomial size and the longest path from the root to the leaf involves removing a single component at a time, the height of Power$(\mathcal{U})$ is polynomial. $\qquad \square$

**Lemma 6.2.** *Power$(\mathcal{U})$ is learnable in the MBP framework.*

*Proof.* This follows from Lemma 6.1 and Theorem 5. $\qquad \square$

**Lemma 6.3.** *For any goal language $\mathcal{G}$, (Power($\mathcal{U}$), $\mathcal{G}$) permits efficient experiment design if (Pairs($\mathcal{U}$), $\mathcal{G}$) permits efficient experiment design.*

*Proof.* Any informative state for a hypothesis in Power($\mathcal{U}$) is an informative state for some hypothesis in Pairs($\mathcal{U}$), and vice versa. Hence, a cover for (Pairs($\mathcal{U}$), $\mathcal{G}$) would be a cover for $(Power(\mathcal{U}), \mathcal{G})$. Consequently, if (Pairs($\mathcal{U}$), $\mathcal{G}$) permits efficient experiment design, then the efficient algorithms EX-PERIMENTDESIGN and INFOACTIONSTATES are directly applicable to (Power($\mathcal{U}$), $\mathcal{G}$). $\square$

**Lemma 6.4.** *For any goal language $\mathcal{G}$, (Power($\mathcal{U}$), $\mathcal{G}$) is learnable in the PLEX framework if (Pairs($\mathcal{U}$), $\mathcal{G}$) permits efficient experiment design and Power($\mathcal{U}$) is adequate for $\mathcal{G}$.*

*Proof.* This follows from Lemmas 6.1 and 6.3, and Theorem 6. $\square$

We now define a hypothesis space that is a concrete member of the family. Let an *action production* $r$ be defined as "act : pre $\rightarrow$ post", where act($r$) is an action and the precondition pre($r$) and postcondition post($r$) are conjunctions of "variable = value" literals.

**Definition 6.1.** *A production $r$ is triggered by a transition $(s, a, s')$ if $s$ satisfies the precondition $pre(r)$ and $a = act(r)$. A production $r$ is consistent with $(s, a, s')$ if either ① $r$ is not triggered by $(s, a, s')$ or ② $s'$ satisfies the post($r$) and all variables not mentioned in post($r$) have the same values in both $s$ and $s'$.*

A production represents the set of all consistent transitions that trigger it. All the variables in pre($r$) must take their specified values in a state to trigger $r$; when $r$ is triggered, post($r$) defines the values in the next state. An example of an action production is "Do : $v_1 = 0, v_2 = 1 \rightarrow v_1 = 2, v_3 = 1$". It is triggered only when the Do action is executed in a state in which $v_1 = 0$ and $v_2 = 1$, and defines the value of $v_1$ to be 2 and $v_3$ to be 1 in the next state, with all other variables staying unchanged.

Let $k$-SAP be the hypothesis space of models represented by a set of action productions (SAP) with no more than $k$ variables per production. If $\mathcal{U}$ is the set of productions, then $|\mathcal{U}| = O\big(|A| \sum_{i=1}^{k} \binom{n}{i}(|D| + 1)^{2i}\big) = O(|A|n^k|D|^{2k})$, because a production can have one of $|A|$ actions, up to $k$ relevant variables figuring on either side of the production, and each variable set to a value in its domain. As $\mathcal{U}$ is of polynomial size, $k$-SAP is an instance of the family of basis action models. Moreover, if Conj is the goal language consisting of all goals that can be expressed as conjunctions of "variable = value" literals, then (Pairs($k$-SAP), Conj) permits efficient experiment design.

**Lemma 6.5.** *($k$-SAP, Conj) is learnable in the PLEX framework if $k$-SAP is adequate for Conj.*

# 7 Conclusion

The main contributions of the paper are the development of the MBP and PLEX frameworks for learning action models and the characterization of sufficient conditions for efficient learning in these frameworks. It also provides results on learning a family of hypothesis spaces that is, in some ways, more general than standard action modeling languages. For example, unlike propositional STRIPS operators, $k$-SAP captures the conditional effects of actions.

While STRIPS-like languages served us well in planning research by creating a common useful platform, they are not designed from the point of view of learnability or planning efficiency. Many domains such as robotics and real-time strategy games are not amenable to such clean and simple action specification languages. This suggests an approach in which the learner considers increasingly complex models as dictated by its planning needs. For example, the model learner might start with small values of $k$ in $k$-SAP and then incrementally increase $k$ until a value is found that is adequate for the goals encountered. In general, this motivates a more comprehensive framework in which planning and learning are tightly integrated, which is the premise of this chapter. Another direction is to investigate better exploration methods that go beyond using optimistic models to include Bayesian and utility-guided optimal exploration.

# 8 Acknowledgments

We thank the reviewers for their helpful feedback. This research is supported by the Army Research Office under grant number W911NF-09-1-0153.

# References

[1] R. Brafman and M. Tennenholtz. R-MAX — A General Polynomial Time Algorithm for Near-Optimal Reinforcement Learning. *Journal of Machine Learning Research*, 3:213–231, 2002.

[2] M. Kearns and L. Valiant. Cryptographic Limitations on Learning Boolean Formulae and Finite Automata. In *Annual ACM Symposium on Theory of Computing*, 1989.

[3] L. Li. *A Unifying Framework for Computational Reinforcement Learning Theory*. PhD thesis, Rutgers University, 2009.

[4] L. Li, M. Littman, and T. Walsh. Knows What It Knows: A Framework for Self-Aware Learning. In *ICML*, 2008.

[5] N. Littlestone. *Mistake Bounds and Logarithmic Linear-Threshold Learning Algorithms*. PhD thesis, U.C. Santa Cruz, 1989.

[6] B. Marthi, S. Russell, and J. Wolfe. Angelic Semantics for High-Level Actions. In *ICAPS*, 2007.

[7] B. K. Natarajan. On Learning Boolean Functions. In *Annual ACM Symposium on Theory of Computing*, 1987.

[8] T. Walsh and M. Littman. Efficient Learning of Action Schemas and Web-Service Descriptions. In *AAAI*, 2008.

